# Empirical Risk Minimization
# with Approximations of Probabilistic Grammars

**Shay B. Cohen**
Language Technologies Institute
School of Computer Science
Carnegie Mellon University
Pittsburgh, PA 15213, USA
scohen@cs.cmu.edu

**Noah A. Smith**
Language Technologies Institute
School of Computer Science
Carnegie Mellon University
Pittsburgh, PA 15213, USA
nasmith@cs.cmu.edu

## Abstract

Probabilistic grammars are generative statistical models that are useful for compositional and sequential structures. We present a framework, reminiscent of structural risk minimization, for empirical risk minimization of the parameters of a fixed probabilistic grammar using the log-loss. We derive sample complexity bounds in this framework that apply both to the supervised setting and the unsupervised setting.

## 1 Introduction

Probabilistic grammars are an important statistical model family used in natural language processing [7], computer vision [16], computational biology [19] and more recently, in human activity analysis [12]. They are commonly estimated using maximum likelihood estimate or variants. Such estimation can be viewed as minimizing empirical risk with the log-loss [21]. The log-loss is not bounded when applied to probabilistic grammars, and that makes it hard to obtain uniform convergence results. Such results would help in deriving sample complexity bounds, that is, bounds on the number of training examples required to obtain accurate estimation.

To overcome this problem, we derive distribution-dependent uniform convergence results for probabilistic grammars. In that sense, our learning framework relates to previous work about learning in a distribution-dependent setting [15] and structural risk minimization [21]. Our work is also related to [8], which discusses the statistical properties of estimation of parsing models in a distribution-free setting. Based on the notion of bounded approximations [1, 9], we define a sequence of increasingly better approximations for probabilistic grammars, which we call "proper approximations." We then derive sample complexity bounds in our framework, for both the supervised case and the unsupervised case.

Our results rely on an exponential decay in probabilities with respect to the length of the derivation (number of derivation steps the grammar takes when generating a structure). This means that most of the probability mass for such a distribution is concentrated on a small number of grammatical derivations. We formalize this notion, and use it in many of our results. For applications involving real-world data of finite size (as in natural language processing, computational biology, and so on), we believe this is a reasonable assumption.

The rest of the paper is organized as follows. §2 gives an overview of probabilistic grammars. §3 gives an overview of the learning setting. §4 presents proper approximations, which are approximate concept spaces that permit the derivation of sample complexity bounds for probabilistic grammars. §5 describes the main sample complexity results. We discuss our results in §6 and conclude in §7.

## 2 Probabilistic Grammars

A probabilistic grammar defines a probability distribution over grammatical derivations generated through a step-by-step process. For example, probabilistic context-free grammars (PCFGs) generate phrase-structure trees by recursively rewriting nonterminal symbols as sequences of "child" symbols according to a fixed set of production rules. Each rewrite of a PCFG is conditionally independent of previous ones given one PCFG state; this Markov property permits efficient inference for the probability distribution defined by the probabilistic grammar.

In this paper, we will assume that any grammatical derivation $z$ fully determines a string $x$, denoted yield($z$). There may be many derivations $z$ for a given string (perhaps infinitely many for some kinds of grammars; we assume that the number of derivations is finite). In general, a probabilistic grammar defines the probability of a grammatical derivation $z$ as:

$$ h_{\boldsymbol{\theta}}(z) \quad = \quad \prod_{k=1}^{K} \prod_{i=1}^{N_k} \theta_{k,i}^{\psi_{k,i}(z)} \quad = \quad \exp \sum_{k=1}^{K} \sum_{i=1}^{N_k} \psi_{k,i}(z) \log \theta_{k,i} \tag{1} $$

$\psi_{k,i}$ is a function that "counts" the number of times the $k$th distribution's $i$th event occurs in the derivation. The $\boldsymbol{\theta}$ are a collection of $K$ multinomials $\langle \boldsymbol{\theta}_1, ..., \boldsymbol{\theta}_K \rangle$, the $k$th of which includes $N_k$ events. We let $N = \sum_{k=1}^{K} N_k$ denote the total number of derivation event types. $D(\boldsymbol{G})$ denotes the set of all possible derivations of $\boldsymbol{G}$. We define $D_x(\boldsymbol{G}) = \{z \in D(\boldsymbol{G}) \mid \text{yield}(z) = x\}$. We let $|x|$ denote the length of the string $x$, and $|z| = \sum_{k=1}^{K} \sum_{i=1}^{N_k} \psi_{k,i}(z)$ denote the "length" (number of event tokens) of the derivation $z$.

Parameter estimation for probabilistic grammars means choosing $\boldsymbol{\theta}$ from complete data ("supervised") or incomplete data ("semi-supervised" or "unsupervised," the latter usually implying that strings $x$ are evidence but all derivations $z$ are missing). We can view parameter estimation as identifying a hypothesis from $\mathcal{H}(\boldsymbol{G}) = \{h_{\boldsymbol{\theta}}(z) \mid \boldsymbol{\theta}\}$ or, equivalently, from $\mathcal{F}(\boldsymbol{G}) = \{-\log h_{\boldsymbol{\theta}}(z) \mid \boldsymbol{\theta}\}$. For simplicity of notation, we assume that there is a fixed grammar and use $\mathcal{H}$ to refer to $\mathcal{H}(\boldsymbol{G})$ and $\mathcal{F}$ to refer to $\mathcal{F}(\boldsymbol{G})$.[1] For every $f_{\boldsymbol{\theta}} \in \mathcal{F}$ we have parameters $\boldsymbol{\theta}$ such that $f_{\boldsymbol{\theta}}(z) = -\sum_{k=1}^{K} \sum_{i=1}^{N_k} \psi_{k,i}(z) \log \theta_{k,i}$.

We will make a few assumptions about $\boldsymbol{G}$ and $\mathbb{P}(z)$, the distribution that generates derivations from $D(\boldsymbol{G})$ (note that $\mathbb{P}$ does not have to be a probabilistic grammar):

- Bounded derivation length: There is an $\alpha \geq 1$ such that, for all $z$, $|z| \leq \alpha|\text{yield}(z)|$. Further, $|z| \geq |x|$.
- Exponential decay of derivations: There is a constant $r < 1$ and a constant $L \geq 0$ such that $\mathbb{P}(z) \leq Lr^{|z|}$.
- Exponential decay of strings: Let $\Lambda(k) = |\{z \in D(\boldsymbol{G}) \mid |z| = k\}|$ be the number derivations of length $k$ in $\boldsymbol{G}$. Taking $r$ as above, then we assume there exists a constant $q < 1$, such that $\Lambda(k)r^k \leq q^k$. This implies that the number of derivations of length $k$ may be exponentially large (e.g., as with many PCFGs), but is bounded by $(q/r)^k$.
- Bounded expectations of rules: There is a $B < \infty$ such that $\mathbb{E}[\psi_{k,i}(z)] \leq B$ for all $k$ and $i$.

We note that, for example, these assumptions must hold for any $\mathbb{P}$ whose support consists of a finite set. These assumptions also hold in many cases when $\mathbb{P}$ itself is a probabilistic grammar. See supplementary material for a note about these assumptions, their empirical justification and the relationship to Tsybakov noise [20, 15].

## 3 The Learning Setting

In the supervised learning setting, a set of grammatical derivations $z_1, \ldots, z_n$ is used to estimate $\boldsymbol{\theta}$, implying a choice of $h \in \mathcal{H}$ that "agrees" with the training data. MLE chooses $h^* \in \mathcal{H}$ to maximize the likelihood of the data:

$$ h^* = \underset{h \in \mathcal{H}}{\operatorname{argmax}} \frac{1}{n} \sum_{i=1}^{n} \log h(z_i) = \underset{h \in \mathcal{H}}{\operatorname{argmin}} \underbrace{\sum_{z \in D(\boldsymbol{G})} \tilde{\mathbb{P}}(z) \left(-\log h(z)\right)}_{R_{\text{emp},n}(-\log h)} \tag{2} $$

As shown, this equates to minimizing the empirical risk, or the expected value of a particular loss function known as log-loss. The expected risk, under $\mathbb{P}$, is the (unknowable) quantity

$$R(-\log h) = \sum_{z \in D(\boldsymbol{G})} \mathbb{P}(z) \left(-\log h(z)\right) = \mathbb{E}_{\mathbb{P}}[-\log h]$$

Showing convergence of the form $\sup_{h \in \mathcal{H}} |R_{\text{emp},n}(-\log h) - R(-\log h)| \underset{n \to \infty}{\longrightarrow} 0$ (in probability), is referred to as double-sided uniform convergence. (We note that $\sup_{h \in \mathcal{H}} |R_{\text{emp},n}(-\log h) - R(-\log h)| = \sup_{f \in \mathcal{F}} |R_{\text{emp},n}(f) - R(f)|$.) This kind of uniform convergence is the driving force in showing that the empirical risk minimizer is *consistent*, i.e., the minimized empirical risk converges to the minimized expected risk. We assume familiarity with the relevant literature about empirical risk minimization; see [21].

## 4  Proper Approximations

The log-loss is unbounded, so that there is no function $F : D(\boldsymbol{G}) \to \mathbb{R}$ such that, $\forall f \in \mathcal{F}$, $\forall z \in D(\boldsymbol{G})$, $f(z) \leq F(z)$; i.e., there is no envelope to uniformly bound $\mathcal{F}$. This makes it difficult to obtain a uniform convergence result of $\sup_{f \in \mathcal{F}} |R_{\text{emp},n}(f) - R(f)|$. Vapnik [21, page 93] shows that we can still get consistency for the maximum likelihood estimator, if we bound from below and above the family of probability distributions at hand.

Instead of making this restriction, which is heavy for probabilistic grammars, we revise the learning model according to well-known results about the convergence of stochastic processes. The revision approximates the concept space using a sequence $\mathcal{F}_1, \mathcal{F}_2, \ldots$ and replaces two-sided uniform convergence with convergence on the sequence of concept spaces. The concept spaces in the sequence vary as a function of the number of samples we have. We next construct the sequence of concept spaces, and in §5 we return to the learning model. Our approximations are based on the concept of *bounded approximations* [1, 9].

Let $\mathcal{F}_m$ (for $m \in \{1, 2, \ldots\}$) be a sequence of concept spaces contained in $\mathcal{F}$. We will require that as $m$ grows larger, $\mathcal{F}_m$ becomes a better approximation of the original concept space $\mathcal{F}$. We say that the sequence "properly approximates" $\mathcal{F}$ if there exists a non-increasing function $\epsilon_{\text{tail}}(m)$ such that $\epsilon_{\text{tail}}(m) \underset{m \to \infty}{\longrightarrow} 0$, a non-increasing function $\epsilon_{\text{bound}}(m)$ such that $\epsilon_{\text{bound}}(m) \underset{m \to \infty}{\longrightarrow} 0$, and an operator $C_m : \mathcal{F} \to \mathcal{F}_m$ such that for all $m$ larger than some $M$:

$$
\begin{aligned}
\text{Containment:} \quad & \mathcal{F}_m \subseteq \mathcal{F} \\
\text{Boundedness:} \quad & \exists K_m \geq 0, \ \forall f \in \mathcal{F}_m, \ \mathbb{E}\left[|f| \times I(|f| \geq K_m)\right] \leq \epsilon_{\text{bound}}(m) \\
\text{Tightness:} \quad & \mathbb{P}\left(\bigcup_{f \in \mathcal{F}} \left\{z \mid C_m(f)(z) - f(z) \geq \epsilon_{\text{tail}}(m)\right\}\right) \leq \epsilon_{\text{tail}}(m)
\end{aligned}
$$

The second requirement bounds the expected values of $\mathcal{F}_m$ on values larger than $K_m$. This is required to obtain uniform convergence results in the revised model [18]. Note that $K_m$ can grow arbitrarily large. The third requirement ensures that our approximation actually converges to the original concept space $\mathcal{F}$. We will show in §4.2 this is actually a well-motivated characterization of convergence for probabilistic grammars in the supervised setting.

We note that a good approximation would have $K_m$ increasing fast as a function of $m$ and $\epsilon_{\text{tail}}(m)$ and $\epsilon_{\text{bound}}(m)$ decreasing fast as a function of $m$. As we will see in §5, we cannot have an arbitrarily fast convergence rate (by, for example, taking a subsequence of $\mathcal{F}_m$), because the size of $K_m$ has a great effect on the number of samples required to obtain accurate estimation.

### 4.1  Constructing Proper Approximations for Probabilistic Grammars

We now focus on constructing proper approximations for probabilistic grammars. We make an assumption about the probabilistic grammar that $\forall k, N_k = 2$. For most common grammar formalisms, this does not change the expressive power: any grammar that can be expressed using $N_k > 2$ can be expressed using a grammar that has $N_k \leq 2$. See supplementary material and [6].

We now construct $\mathcal{F}_m$. For each $f \in \mathcal{F}$ we define the transformation $T(f, \gamma)$ that shifts every $\theta_k = \langle \theta_{k,1}, \theta_{k,2} \rangle$ in the probabilistic grammar by $\gamma$:

$$\langle \theta_{k,1}, \theta_{k,2} \rangle \leftarrow \begin{cases} \langle \gamma, & 1 - \gamma \rangle & \text{if} & \theta_{k,1} & < & \gamma \\ \langle 1 - \gamma, & \gamma \rangle & \text{if} & \theta_{k,1} & > & 1 - \gamma \\ \langle \theta_{k,1}, & \theta_{k,2} \rangle & \text{otherwise} \end{cases} \tag{3}$$

Note that $T(f, \gamma) \in \mathcal{F}$ for any $\gamma \le 1/2$. Fix a constant $p > 1$. For each $m \in \mathbb{N}$, define $\mathcal{F}_m = \{T(f, m^{-p}) \mid f \in \mathcal{F}\}$.

**Proposition 4.1.** *There exists a constant* $\beta = \beta(L, q, p, N) > 0$ *such that* $\mathcal{F}_m$ *has the boundedness property with* $K_m = pN \log^3 m$ *and* $\epsilon_{\mathrm{bound}}(m) = m^{-\beta \log m}$.

*Proof.* Let $f \in \mathcal{F}_m$. Let $\mathcal{Z}(m) = \{z \mid |z| \le \log^2 m\}$. Then, for all $z \in \mathcal{Z}(m)$ we have $f(z) = -\sum_{i,k} \psi(k, i) \log \theta_{k,i} \le \sum_{i,k} \psi(k, i)(p \log m) \le pN \log^3 m = K_m$, where the first inequality follows from $f \in \mathcal{F}_m$ $(\theta_{k,i} \ge m^{-p})$ and the second from $|z| \le \log^2 m$. In addition, from the requirements on $\mathbb{P}$ we have:

$$\mathbb{E}\left[|f| \times I(|f| \ge K_m)\right] \le pN \log m \left(\sum_{k > \log^2 m} L\Lambda(k) r^k k\right) \le \kappa \log m \left(q^{\log^2 m}\right)$$

for some constant $\kappa > 0$. Finally, for some $\beta(L, q, p, N) = \beta > 0$ and some constant $M$, if $m > M$ then $\kappa \log m \left(q^{\log^2 m}\right) \le m^{-\beta \log m}$. $\square$

We show now that $\mathcal{F}_m$ is *tight* with respect to $\mathcal{F}$ with $\epsilon_{\mathrm{tail}}(m) = \dfrac{N \log^2 m}{m^p - 1}$:

**Proposition 4.2.** *There exists an* $M$ *such that for any* $m > M$ *we have:*
$$\mathbb{P}\left(\bigcup_{f \in \mathcal{F}}\{z \mid C_m(f)(z) - f(z) \ge \epsilon_{\mathrm{tail}}(m)\}\right) \le \epsilon_{\mathrm{tail}}(m) \ \text{for} \ \epsilon_{\mathrm{tail}}(m) = \dfrac{N \log^2 m}{m^p - 1} \ \text{and}$$
$C_m(f) = T(f, m^{-p})$.

*Proof.* See supplementary material. $\square$

We now have proper approximations for probabilistic grammars. From this point, we use $\mathcal{F}_m$ to denote the proper approximation constructed for $\mathbf{G}$. We use $\epsilon_{\mathrm{bound}}(m)$ and $\epsilon_{\mathrm{tail}}(m)$ as in Proposition 4.1 and Proposition 4.2, and assume that $p > 1$ is fixed, for the rest of the paper.

### 4.2 Asymptotic Empirical Risk Minimization

It would be compelling to know that the empirical risk minimizer over $\mathcal{F}_n$ is an *asymptotic empirical risk minimizer* (in the log-loss case, this means it converges to the maximum likelihood estimate). As a conclusion to this section about proper approximations, we motivate the three requirements that we posed on proper approximations by showing that this is indeed true. We now unify $n$, the number of samples, and $m$, the index of the approximation of the concept space $\mathcal{F}$. Let $f_n^*$ be the minimizer of the empirical risk over $\mathcal{F}$, $(f_n^* = \mathrm{argmin}_{f \in \mathcal{F}} R_{\mathrm{emp},n}(f))$ and let $g_n$ be the minimizer of the empirical risk over $\mathcal{F}_n$ $(g_n = \mathrm{argmin}_{f \in \mathcal{F}_n} R_{\mathrm{emp},n}(f))$.

Let $D = \{z_1, ..., z_n\}$ be a sample from $\mathbb{P}(z)$. The operator $(g_n =) \mathrm{argmin}_{f \in \mathcal{F}_n} R_{\mathrm{emp},n}(f)$ is an asymptotic empirical risk minimizer if $\mathbb{E}[R_{\mathrm{emp},n}(g_n) - R_{\mathrm{emp},n}(f_n^*)] \to 0$. Then, we have the following:

**Proposition 4.3.** *Let* $D = \{z_1, ..., z_n\}$ *be a sample of derivations for* $\mathbf{G}$. *Then* $g_n = \mathrm{argmin}_{f \in \mathcal{F}_n} R_{\mathrm{emp},n}(f)$ *is an asymptotic empirical risk minimizer.*

**Lemma 4.4.** *Denote by* $\mathcal{Z}_{\epsilon,n}$ *the set* $\bigcup_{f \in \mathcal{F}}\{z \mid C_n(f)(z) - f(z) \ge \epsilon\}$. *Denote by* $A_{\epsilon,n}$ *the event "one of* $z_i \in D$ *is in* $\mathcal{Z}_{\epsilon,n}$." *Then if* $\mathcal{F}_n$ *properly approximates* $\mathcal{F}$ *then:*

$$\mathbb{E}\left[R_{\mathrm{emp},n}(g_n) - R_{\mathrm{emp},n}(f_n^*)\right] \tag{4}$$
$$\le \left|\mathbb{E}\left[R_{\mathrm{emp},n}(C_n(f_n^*)) \mid A_{\epsilon,n}\right]\right|\mathbb{P}(A_{\epsilon,n}) + \left|\mathbb{E}\left[R_{\mathrm{emp},n}(f_n^*) \mid A_{\epsilon,n}\right]\right|\mathbb{P}(A_{\epsilon,n}) + \epsilon_{\mathrm{tail}}(n)$$

*where the expectations are taken with respect to the dataset* $D$. (See the supplementary material for a proof.)

*Proof of Proposition 4.3.* Let $f_0 \in \mathcal{F}$ be the concept that puts uniform weights over $\boldsymbol{\theta}$, i.e., $\boldsymbol{\theta}_k = \langle \frac{1}{2}, \frac{1}{2} \rangle$ for all $k$. Note that $|\mathbb{E}[R_{\mathrm{emp},n}(f_n^*) \mid A_{\epsilon,n}]|\mathbb{P}(A_{\epsilon,n})$

$$\leq |\mathbb{E}[R_{\mathrm{emp},n}(f_0) \mid A_{\epsilon,n}]|\mathbb{P}(A_{\epsilon,n}) = \frac{\log 2}{n} \sum_{l=1}^n \sum_{k,i} \mathbb{E}[\psi_{k,i}(z_l) \mid A_{\epsilon,n}]\mathbb{P}(A_{\epsilon,n}) \tag{5}$$

Let $A_{j,\epsilon,n}$ for $j \in \{1, \ldots, n\}$ be the event "$z_j \in \mathcal{Z}_{\epsilon,n}$". Then $A_{\epsilon,n} = \bigcup_j A_{j,\epsilon,n}$. We have that:

$$\mathbb{E}[\psi_{k,i}(z_l) \mid A_{\epsilon,n}]\mathbb{P}(A_{\epsilon,n}) \leq \sum_j \sum_{z_l} \mathbb{P}(z_l, A_{j,\epsilon,n})|z_l| \tag{6}$$

$$\leq \sum_{j \neq l} \sum_{z_l} \mathbb{P}(z_l)\mathbb{P}(A_{j,\epsilon,n})|z_l| + \sum_{z_l} \mathbb{P}(z_l, A_{l,\epsilon,n})|z_l| \tag{7}$$

$$\leq \left(\sum_{j \neq l} \mathbb{P}(A_{j,\epsilon,n})\right) B + \mathbb{E}[\psi_{k,i}(z) \mid z \in \mathcal{Z}_{\epsilon,n}]\mathbb{P}(z \in \mathcal{Z}_{\epsilon,n}) \tag{8}$$

$$\leq (n-1)B\mathbb{P}(z \in \mathcal{Z}_{\epsilon,n}) + \mathbb{E}[\psi_{k,i}(z) \mid z \in \mathcal{Z}_{\epsilon,n}]\mathbb{P}(z \in \mathcal{Z}_{\epsilon,n}) \tag{9}$$

where Eq. 7 comes from $z_l$ being independent and $B$ is the constant from §2. Therefore, we have:

$$\frac{1}{n}\sum_{l=1}^n \sum_{k,i} \mathbb{E}[\psi_{k,i}(z_l) \mid A_{\epsilon,n}]\mathbb{P}(A_{\epsilon,n}) \leq \sum_{k,i} \left(\mathbb{E}[\psi_{k,i}(z) \mid z \in \mathcal{Z}_{\epsilon,n}]\mathbb{P}(z \in \mathcal{Z}_{\epsilon,n}) + n^2 B\mathbb{P}(z \in \mathcal{Z}_{\epsilon,n})\right)$$

$$\tag{10}$$

From the construction of our proper approximations (Proposition 4.2), we know that only derivations of length $\log^2 n$ or greater can be in $\mathcal{Z}_{\epsilon,n}$. Therefore:

$$\mathbb{E}[\psi_{k,i} \mid \mathcal{Z}_{\epsilon,n}]\mathbb{P}(\mathcal{Z}_{\epsilon,n}) \leq \sum_{z:|z|>\log^2 n} \mathbb{P}(z)\psi_{k,i}(z) \leq \sum_{l>\log^2 n}^{\infty} L\Lambda(l)r^l l \leq \kappa q^{\log^2 n} = o(1) \tag{11}$$

where $\kappa > 0$ is a constant. Similarly, we have $\mathbb{P}(z \in \mathcal{Z}_{\epsilon,n}) = o(n^{-2})$. This means that $|\mathbb{E}[R_{\mathrm{emp},n}(f_n^*) \mid A_{\epsilon,n}]|\mathbb{P}(A_{\epsilon,n}) \xrightarrow[n\to\infty]{} 0$. In addition, it can be shown $|\mathbb{E}[R_{\mathrm{emp},n}(C_n(f_n^*)) \mid A_{\epsilon,n}]|\mathbb{P}(A_{\epsilon,n}) \xrightarrow[n\to\infty]{} 0$ using the same proof technique we used above, while relying on the fact that $C_n(f_n^*) \in \mathcal{F}_n$, and therefore $C_n(f_n^*)(z) \leq pN|z|\log n$. $\qquad\square$

## 5  Sample Complexity Results

We now give our main sample complexity results for probabilistic grammars. These results hinge on the convergence of $\sup_{f \in \mathcal{F}_n} |R_{\mathrm{emp},n}(f) - R(f)|$. The rate of this convergence can be fast, if the *covering numbers* for $\mathcal{F}_n$ do not grow too fast.

We next give a brief overview of covering numbers. A cover gives a way to reduce a class of functions to a much smaller (finite, in fact) representative class such that each function in the original class is represented using a function in the smaller class. Let $\mathcal{G}$ be a class of functions. Let $d(f, g)$ be a distance measure between two functions $f, g$ from $\mathcal{G}$. An $\epsilon$-cover is a subset of $\mathcal{G}$, denoted by $\mathcal{G}'$, such that for every $f \in \mathcal{G}$ there exists an $f' \in \mathcal{G}'$ such that $d(f, f') < \epsilon$. The covering number $\mathcal{N}(\epsilon, \mathcal{G}, d)$ is the size of the smallest $\epsilon$-cover of $\mathcal{G}$ using with respect to the distance measure $d$.

We will be interested in a specific distance measure that is dependent on the empirical distribution $\tilde{\mathbb{P}}$ that describes the data $z_1, ..., z_n$. Let $f, g \in \mathcal{G}$. We will use:

$$d^{\tilde{\mathbb{P}}}(f, g) \;=\; \mathbb{E}_{\tilde{\mathbb{P}}}[|f - g|] \;=\; \sum_{z \in D(\boldsymbol{G})} |f(z) - g(z)|\, \tilde{\mathbb{P}}(z) \;=\; \frac{1}{n}\sum_{i=1}^n |f(z_i) - g(z_i)| \tag{12}$$

Instead of using $\mathcal{N}(\epsilon, \mathcal{G}, d^{\tilde{\mathbb{P}}})$ directly, we are going to bound this quantity with $\mathcal{N}(\epsilon, \mathcal{G}) = \sup_{\tilde{\mathbb{P}}} \mathcal{N}(\epsilon, \mathcal{G}, d^{\tilde{\mathbb{P}}})$, where we consider all possible samples (yielding $\tilde{\mathbb{P}}$). The following is the key result about the connection between covering numbers and the double-sided convergence of the empirical process $\sup_{f \in \mathcal{F}_n} |R_{\mathrm{emp},n}(f) - R(f)|$ as $n \to \infty$:

**Lemma 5.1.** *Let $\mathcal{F}_n$ be a permissible class[2] of functions such that for every $f \in \mathcal{F}_n$ we have $\mathbb{E}[|f|I(|f| \leq K_n)] \leq \epsilon_{\mathrm{bound}}(n)$. Let $\mathcal{F}_{\mathrm{truncated},n} = \{f \times I(f \leq K_n) \mid f \in \mathcal{F}_m\}$, i.e., the set of*

*functions from $\mathcal{F}_n$ after being truncated by $K_n$. Then for $\epsilon > 0$ we have,*

$$\mathbb{P}\left(\sup_{f \in \mathcal{F}_n} |R_{\mathrm{emp},n}(f) - R(f)| > 2\epsilon\right) \leq 8\mathcal{N}(\epsilon/8, \mathcal{F}_{\mathrm{truncated},n})\exp\left(-\frac{1}{128}n\epsilon^2/K_n^2\right) + 2\epsilon_{\mathrm{bound}}(n)/\epsilon$$

*provided $n \geq K_n^2/4\epsilon^2$.*

*Proof.* See [18] (chapter 2, pages 30–31). See supplementary material for an explanation. $\square$

Covering numbers are rather complex combinatorial quantities that are hard to compute directly. Fortunately, they can be bounded by using the pseudo dimension [3], a generalization of VC dimension for real functions. In the case of our "binomialized" probabilistic grammars, the pseudo dimension of $\mathcal{F}_n$ is bounded by $N$, because we have $\mathcal{F}_n \subseteq \mathcal{F}$, and the functions in $\mathcal{F}$ are linear with $N$ parameters. Hence, $\mathcal{F}_{\mathrm{truncated},n}$ has also pseudo dimension that is at most $N$. We have:

**Lemma 5.2.** *(From [18, 13].) Let $\mathcal{F}_n$ be the proper approximations for probabilistic grammars, for any $0 < \epsilon < K_n$ we have:*

$$\mathcal{N}(\epsilon, \mathcal{F}_{\mathrm{truncated},n}) < 2\left(\frac{2eK_n}{\epsilon}\log\frac{2eK_n}{\epsilon}\right)^N \tag{13}$$

## 5.1 Supervised Case

Lemmas 5.1 and 5.2 can be combined to get our main sample complexity result:

**Theorem 5.3.** *Let $G$ be a grammar. Let $\mathcal{F}_n$ be a proper approximation for the corresponding family of probabilistic grammars. Let $\mathbb{P}(x, z)$ be a distribution over derivations that satisfies the requirements in §2. Let $z_1, ..., z_n$ be a sample of derivations. Then there exists a constant $\beta(L, q, p, N)$ and constant $M$ such that for any $0 < \delta < 1$ and $0 < \epsilon < 1$ and any $n > M$ and if*

$$n \geq \max\left\{\frac{128K_n^2}{\epsilon^2}\left(2N\log(16eK_n/\epsilon) + \log\frac{32}{\delta}\right), \frac{\log 4/\delta + \log 1/\epsilon}{\beta(L, q, p, N)}\right\} \tag{14}$$

*then we have*

$$\mathbb{P}\left(\sup_{f \in \mathcal{F}_n} |R_{\mathrm{emp},n}(f) - R(f)| \leq 2\epsilon\right) \geq 1 - \delta \tag{15}$$

*where $K_n = pN\log^3 n$.*

*Proof.* Omitted for space. $\beta(L, q, p, N)$ is the constant from Proposition 4.1. The proof is based on simple algebraic manipulation of the right side of Eq. 13 while relying on Lemma 5.2. $\square$

## 5.2 Unsupervised Case

In the unsupervised setting, we have $n$ *yields* of derivations from the grammar, $x_1, ..., x_n$, and our goal again is to identify grammar parameters $\boldsymbol{\theta}$ from these yields. Our concept classes are now the sets of log marginalized distributions from $\mathcal{F}_n$. For each $f_{\boldsymbol{\theta}} \in \mathcal{F}_n$, we define $f'_{\boldsymbol{\theta}}$ as:

$$f'_{\boldsymbol{\theta}}(x) = -\log\sum_{z \in D_x(\boldsymbol{G})}\exp(-f_{\boldsymbol{\theta}}(z)) = -\log\sum_{z \in D_x(\boldsymbol{G})}\exp\left(\sum_{k=1}^{K}\sum_{i=1}^{N_k}\psi_{i,k}(z)\boldsymbol{\theta}_{i,k}\right) \tag{16}$$

We denote the set of $\{f'_{\boldsymbol{\theta}}\}$ by $\mathcal{F}'_n$. We define analogously $\mathcal{F}'$. Note that we also need to define the operator $C'_n(f')$ as a first step towards defining $\mathcal{F}'_n$ as proper approximations (for $\mathcal{F}'$) in the unsupervised setting. Let $f' \in \mathcal{F}'$. Let $f$ be the concept in $\mathcal{F}$ such that $f'(x) = \sum_z f(z, x)$. Then we define $C'_n(f')(x) = \sum_z C_n(f)(x, z)$.

It is not immediate to show that $\mathcal{F}'_n$ is a proper approximation for $\mathcal{F}'$. It is not hard to show that the boundedness property is satisfied with the same $K_n$ and the same form of $\epsilon_{\mathrm{bound}}(n)$ as in Proposition 4.1 (we would have $\epsilon'_{\mathrm{bound}}(m) = m^{-\beta'\log m}$ for some $\beta'(L, q, p, N) = \beta' > 0$). This relies on the property of bounded derivation length of $\mathbb{P}$. See the supplementary material for a proof. The following result shows that we have tightness as well:

**Proposition 5.4.** *There exists an $M$ such that for any $n > M$ we have:* $\mathbb{P}\left(\bigcup_{f'\in\mathcal{F}'}\{x \mid C'_n(f')(x) - f'(x) \geq \epsilon_{\text{tail}}(n)\}\right) \leq \epsilon_{\text{tail}}(n)$ *for* $\epsilon_{\text{tail}}(n) = \dfrac{N\log^2 n}{n^p - 1}$ *and the operator* $C'_n(f)$ *as defined above.*

**Utility Lemma 5.5.** *For $a_i, b_i \geq 0$, if $-\log\sum_i a_i + \log\sum_i b_i \geq \epsilon$ then there exists an $i$ such that $-\log a_i + \log b_i \geq \epsilon$.*

*Sketch of proof of Proposition 5.4.* From Utility Lemma 5.5 we have:

$$\mathbb{P}\left(\bigcup_{f'\in\mathcal{F}'}\{x \mid C'_n(f')(x) - f'(x) \geq \epsilon_{\text{tail}}(n)\}\right) \leq \mathbb{P}\left(\bigcup_{f\in\mathcal{F}}\{x \mid \exists z C_n(f)(z) - f(z) \geq \epsilon_{\text{tail}}(n)\}\right) \tag{17}$$

Define $\mathcal{X}(n)$ to be all $x$ such that there exists a $z$ with $\text{yield}(z) = x$ and $|z| \geq \log^2 n$. From the proof of Proposition 4.2 and the requirements on $\mathbb{P}$, we know that there exists an $\alpha \geq 1$ such that

$$\mathbb{P}\left(\bigcup_{f\in\mathcal{F}}\{x \mid \exists z \text{ s.t. } C_n(f)(z) - f(z) \geq \epsilon_{\text{tail}}(n)\}\right) \leq \sum_{x\in\mathcal{X}(n)}\mathbb{P}(x)$$

$$\leq \sum_{x:|x|\geq\log^2 n/\alpha}\mathbb{P}(x) \leq \sum_{k=\lfloor\log^2 n/\alpha\rfloor}^{\infty}L\Lambda(k)r^k \leq \epsilon_{\text{tail}}(n) \tag{18}$$

where the last inequality happens for some $n$ larger than a fixed $M$. $\square$

Computing either the covering number or the pseudo dimension of $\mathcal{F}'_n$ is a hard task, because the function in the classes includes the "log-sum-exp." In [9], Dasgupta overcomes this problem for Bayesian networks with fixed structure by giving a bound on the covering number for (his respective) $\mathcal{F}'$ that depends on the covering number of $\mathcal{F}$.

Unfortunately, we cannot fully adopt this approach, because the derivations of a probabilistic grammar can be arbitrarily large. We overcome this problem using the following restriction. We assume that $|D_x(\mathbf{G})| < d(n)$, where $d$ is a function mapping $n$, the size of our sample, to a real number. The more samples we have, the more permissive (for large derivation set) the grammar can be. On the other hand, the more accuracy we desire, the more restricted we are in choosing grammars that have a large derivation set. We refer to this restriction as the "derivational condition." With the derivational condition, we can show the following result:

**Proposition 5.6.** (Hidden Variable Rule for Probabilistic Grammars) *Under the derivational condition, $\mathcal{N}(\epsilon, \mathcal{F}'_{\text{truncated},n}) \leq \mathcal{N}(\epsilon/d(n), \mathcal{F}_{\text{truncated},n})$.*

The proof of Proposition 5.6 is almost identical to the proof of the hidden variable rule in [9]. For the unsupervised case, then, we get the following sample complexity result:

**Theorem 5.7.** *Let $\mathbf{G}$ be a grammar. Let $\mathcal{F}'_n$ be a proper approximation for the corresponding family of probabilistic grammars. Let $\mathbb{P}(x, z)$ be a distribution over derivations that satisfies the requirements in §2. Let $x_1, ..., x_n$ be a sample of strings from $\mathbb{P}(x)$. Then there exists a constant $\beta'(L, q, p, N)$ and constant $M$ such that for any $0 < \delta < 1$ and $0 < \epsilon < 1$ and any $n > M$ and if*

$$n \geq \max\left\{\frac{128K_n^2}{\epsilon^2}\left(2N\log(16eK_n d(n)/\epsilon) + \log\frac{32}{\delta}\right), \frac{\log 4/\delta + \log 1/\epsilon}{\beta'(L, q, p, N)}\right\} \tag{19}$$

*and $|D_x(\mathbf{G})| < d(n)$, we have that*

$$\mathbb{P}\left(\sup_{f\in\mathcal{F}'_n}|R_{\text{emp},n}(f) - R(f)| \leq 2\epsilon\right) \geq 1 - \delta \tag{20}$$

*where $K_n = pN\log^3 n$.*

For this sample complexity bound to be non-trivial, for example, we can restrict $D_x(\mathbf{G})$, through $d(n)$, to have a polynomial size in the number of our samples. Enlarging $d(n)$ is possible even to an exponential function of $n^\rho$ for $\rho < 1$, e.g. $d(n) = 2^{\sqrt{n}}$.

| criterion | as $K_n$ increases ... | as $d(n)$ increases ... | as $p$ increases ... |
|---|---|---|---|
| tightness of proper approximation | improves | no effect | improves |
| sample complexity bound | degrades | degrades | degrades |

Table 1: Trade-off between quantities in our learning model and effectiveness of different criteria. $d(n)$ is the function that gives the derivational condition, i.e., $|D_x(\boldsymbol{G})| \leq d(n)$.

## 6   Discussion

Our framework can be specialized to improve the two main criteria that have a trade-off: the tightness of the proper approximation and the sample complexity. For example, we can improve the tightness of our proper approximations by taking a subsequence of $\mathcal{F}_n$. However, this will make the sample complexity bound degrade, because $K_n$ will grow faster. Table 1 gives the different trade-offs between parameters in our model and the effectiveness of learning. In general, we would want the derivational condition to be removed (choose $d(n) = \infty$, or at least allow $d(n) = \Omega(t^n)$ for some $t$, for small samples), but in that case our sample complexity bounds become trivial.

In the supervised case, our result states that the number of samples we require (as an upper bound) grows mostly because of a term that behaves $O(N^3 \log N)$ (for a fixed $\delta$ and $\epsilon$). If our grammar, for example, is a PCFG, then $N$ depends on the total number of rules. When the PCFG is in Chomsky normal form and *lexicalized* [10, 7], then $N$ grows by an order of $V^2$, where $V$ is the vocabulary size. This means that the bound grows by an order of $O(V^6 \log V)$. This is consistent with conventional wisdom that lexicalized grammars require much more data for accurate learning.

The dependence of the bound on $N$ suggests that it is easier to learn models with a smaller grammar size. This may help explain the success of recent advances in supervised parsing [4, 22, 17] that have "coarse" models (with a much smaller size of nontermimals) as a first pass. Those models are easier to learn and require less data to be accurate, and can serve as base models for later phases.

The sample complexity bound for the unsupervised case suggests that we need $\log d(n)$ times as much data to achieve estimates as good as those for supervised learning. Interestingly, with unsupervised grammar learning, available training sentences longer than a maximum length (e.g., 10) are often ignored; see [14].

We note that sample complexity is not the only measure for the complexity of estimating probabilistic grammars. In the unsupervised setting, for example, the computational complexity of ERM is NP hard for PCFGs [5] or probabilistic automata [2].

## 7   Conclusion

We presented a framework for learning the parameters of a probabilistic grammar under the log-loss and derived sample complexity bounds for it. We motivated this framework by showing that the empirical risk minimizer for our approximate framework is an asymptotic empirical risk minimizer. Our framework uses a sequence of approximations to a family of probabilistic grammars, which improves as we have more data, to give distribution dependent sample complexity bounds in the supervised and unsupervised settings.

## Acknowledgements

We thank the anonymous reviewers for their comments and Avrim Blum, Steve Hanneke, and Dan Roth for useful conversations. This research was supported by NSF grant IIS-0915187.

## Footnotes

[1]Learning the *rules* in a grammar is another important problem that has received much attention [11].

[2] The "permissible class" requirement is a mild regularity condition about measurability that holds for proper approximations. We refer the reader to [18] for more details.

## References

[1] N. Abe, J. Takeuchi, and M. Warmuth. Polynomial learnability of probabilistic concepts with respect to the Kullback-Leiber divergence. In *ACM Conference on Computational Learning Theory*, 1990.

[2] N. Abe and M. Warmuth. On the computational complexity of approximating distributions by probabilistic automata. *Machine Learning*, 2:205–260, 1992.

[3] M. Anthony and P. L. Bartlett. *Neural Network Learning: Theoretical Foundations*. Cambridge University Press, 1999.

[4] E. Charniak and M. Johnson. Coarse-to-fine $n$-best parsing and maxent discriminative reranking. In *Proc. of ACL*, 2005.

[5] S. B. Cohen and N. A. Smith. Viterbi training for PCFGs: Hardness results and competitiveness of uniform initialization. In *Proceedings of ACL*, 2010.

[6] S. B. Cohen and N. A. Smith. Empirical risk minimization for probabilistic grammars: Sample complexity and hardness of learning, in preparation.

[7] M. Collins. Head-driven statistical models for natural language processing. *Computational Linguistics*, 29:589–637, 2003.

[8] M. Collins. Parameter estimation for statistical parsing models: theory and practice of distribution-free methods. *Text, Speech and Language Technology (new developments in parsing technology)*, pages 19–55, 2004.

[9] S. Dasgupta. The sample complexity of learning fixed-structure Bayesian networks. *Machine Learning*, 29(2-3):165–180, 1997.

[10] J. Eisner. Three new probabilistic models for dependency parsing: An exploration. In *Proc. of COLING*, 1996.

[11] E. M. Gold. Language identification in the limit. *Information and Control*, 10(5):447–474, 1967.

[12] G. Guerra and Y. Aloimonos. Discovering a language for human activity. In *AAAI Workshop on Anticipation in Cognitive Systems*, 2005.

[13] D. Haussler. Decision-theoretic generalizations of the PAC model for neural net and other learning applications. *Information and Computation*, 100:78–150, 1992.

[14] D. Klein and C. D. Manning. Corpus-based induction of syntactic structure: Models of dependency and constituency. In *Proc. of ACL*, 2004.

[15] V. Koltchinskii. Local Rademacher complexities and oracle inequalities in risk minimization. *The Annals of Statistics*, 34(6):2593–2656, 2006.

[16] L. Lin, T. Wu, J. Porway, and Z. Xu. A stochastic graph grammar for compositional object representation and recognition. *Pattern Recognition*, 8, 2009.

[17] S. Petrov and D. Klein. Improved inference for unlexicalized parsing. In *Proc. of HLT-NAACL*, 2007.

[18] D. Pollard. *Convergence of Stochastic Processes*. New York: Springer-Verlag, 1984.

[19] Y. Sakakibara, M. Brown, R. Hughey, S. Mian, K. Sjölander, R. C. Underwood, and D. Haussler. Stochastic context-free grammars for tRNA modeling. *Nucleic Acids Research*, 22, 1994.

[20] A. Tsybakov. Optimal aggregation of classifiers in statistical learning. *The Annals of Statistics*, 32(1):135–166, 2004.

[21] V. N. Vapnik. *Statistical Learning Theory*. Wiley-Interscience, 1998.

[22] D. Weiss and B. Taskar. Structured prediction cascades. In *Proceedings of AISTATS*, 2010.

